# McRank: Learning to Rank Using Multiple Classification and Gradient Boosting

**Ping Li** *
Dept. of Statistical Science
Cornell University
pingli@cornell.edu

**Christopher J.C. Burges**
Microsoft Research
Microsoft Corporation
cburges@microsoft.com

**Qiang Wu**
Microsoft Research
Microsoft Corporation
qiangwu@microsoft.com

## Abstract

We cast the ranking problem as (1) multiple classification ("Mc") (2) multiple ordinal classification, which lead to computationally tractable learning algorithms for relevance ranking in Web search. We consider the DCG criterion (discounted cumulative gain), a standard quality measure in information retrieval. Our approach is motivated by the fact that perfect classifications result in perfect DCG scores and the DCG errors are bounded by classification errors. We propose using the *Expected Relevance* to convert class probabilities into ranking scores. The class probabilities are learned using a gradient boosting tree algorithm. Evaluations on large-scale datasets show that our approach can improve *LambdaRank* [5] and the regressions-based ranker [6], in terms of the (normalized) DCG scores. An efficient implementation of the boosting tree algorithm is also presented.

## 1 Introduction

The general *ranking* problem has widespread applications including commercial search engines and recommender systems. We develop *McRank*, a computationally tractable learning algorithm for the general ranking problem; and we present our approach in the context of ranking in Web search.

For a given user input query, a commercial search engine returns many pages of URLs, in an order determined by the underlying proprietary ranking algorithm. The quality of the returned results are largely evaluated on the URLs displayed in the very first page. The type of ranking problem in this study is sometimes referred to as *dynamic ranking* (or simply, just *ranking*), because the URLs are dynamically ranked (in real-time) according to the specific user input query. This is different from the query-independent *static ranking* based on, for example, "page rank" [3] or "authorities and hubs" [12], which may, at least conceptually, serve as an important "feature" for dynamic ranking or to guide the generation of a list of URLs fed to the dynamic ranker.

There are two main categories of ranking algorithms. A popular scheme is based on learning pairwise preferences, including *RankNet* [4], *LambdaRank* [5], *RankSVM* [11], *RankBoost* [7], *GBRank* [14], and *FRank* [13]. Both *LambdaRank* and *RankNet* used neural nets.[1] *RankNet* used a cross-entropy type of loss function and *LambdaRank* used a gradient based on NDCG smoothed by the RankNet loss. Another scheme is based on regression [6]. [6] considered the DCG measure (discounted cumulative gain) [10] and showed that the DCG errors are bounded by regression errors.

In this study, we also consider the DCG measure. From the definition of DCG, it appears more direct to cast the ranking problem as multiple classification ("Mc") as opposed to regression. In order to convert classification results into ranking scores, we propose a simple and stable mechanism by using the *Expected Relevance*. Our evaluations on large-scale datasets demonstrate the superiority of the classification-based ranker (*McRank*) over both the regression-based and pair-based schemes.

## 2 Discounted Cumulative Gain (DCG)

For an input query, the ranker returns $n$ ordered URLs. Suppose the URLs fed to the ranker are originally ordered $\{1, 2, 3, ..., n\}$. The ranker will output a permutation mapping $\pi : \{1, 2, 3, ..., n\} \rightarrow \{1, 2, 3, ..., n\}$. We denote the inverse mapping by $\sigma_i = \sigma(i) = \pi^{-1}(i)$.

The DCG score is computed from the relevance levels of the $n$ URLs as

$$\text{DCG} = \sum_{i=1}^{n} c_{[i]} \left(2^{y_{\sigma_i}} - 1\right) = \sum_{i=1}^{n} c_{[\pi_i]} \left(2^{y_i} - 1\right), \tag{1}$$

where $[i]$ is the rank order, and $y_i \in \{0, 1, 2, 3, 4\}$ is the *relevance* level of the $i$th URL in the original (pre-ranked) order. $y_i = 4$ corresponds to a "perfect" relevance and $y_i = 0$ corresponds to a "poor" relevance. For generating training datasets, human judges have manually labeled a large number of queries and URLs. In this study, we assume these labels are "gold-standard."

In the definition of DCG, $c_{[i]}$, which is a non-increasing function of $i$, is typically set as

$$c_{[i]} = \frac{1}{\log(1+i)}, \qquad \text{if } i \le L, \qquad \text{and } c_{[i]} = 0, \text{ if } i > L, \tag{2}$$

where $L$ is the "truncation level" and is typically set to be $L = 10$, to reflect the fact that the search quality of commercial search engines is mainly determined by the URLs displayed in the first page.

Suppose a dataset contains $N_Q$ queries. It is a common practice to normalize the DCG score for each query and report the normalized DCG ("NDCG") score averaged over all queries. In other words, the NDCG for the $j$th query ($\text{NDCG}_j$) and the final NDCG of the dataset ($\text{NDCG}_F$) are

$$\text{NDCG}_j = \frac{\text{DCG}_j}{\text{DCG}_{j,g}}, \qquad \text{NDCG}_F = \frac{1}{N_Q} \sum_{j=1}^{N_Q} \text{NDCG}_j, \tag{3}$$

where $\text{DCG}_{j,g}$ is the maximum possible (or "gold standard") DCG score of the $j$th query.

# 3 Learning to Rank Using Classification

The definition of DCG suggests that we can cast the ranking problem naturally as multiple classification (i.e., $K = 5$ classes), because obviously perfect classifications will lead to perfect DCG scores. While the DCG criterion is non-convex and non-smooth, classification is very well-studied.

We should mention that one does not really need perfect classifications in order to produce perfect DCG scores. For example, suppose within a query, the URLs are all labeled level 1 or higher. If an algorithm always classifies the URLs one level lower (i.e., URLs labeled level 4 are classified as level 3, and so on), we still have the perfect DCG score but the classification "error" is $100\%$. This phenomenon to an extent, may provide some "safety cushion" for casting ranking as classification.

[6] cast ranking as regression and showed that the DCG errors are bounded by regression errors. It appears to us that the regression-based approach is less direct and possibly also less accurate than our classification-based proposal. For example, it is well-known that, although one can use regression for classification, it is often better to use logistic regression especially for multiple classification [8].

## 3.1 Bounding DCG Errors by Classification Errors

Following [6, Theorem 2], we show that the DCG errors can be bounded by classification errors.

For a permutation mapping $\pi$, the error is $\text{DCG}_g$ - $\text{DCG}_\pi$. One simple way to obtain the perfect $\text{DCG}_g$ is to rank the URLs directly according to the gold-standard relevance levels. That is, all URLs with relevance level $k+1$ are ranked higher than those with relevance level $\le k$; and the URLs with the same relevance levels are arbitrarily ranked without affecting $\text{DCG}_g$. We denote the corresponding permutation mapping also by $g$.

**Lemma 1** *Given $n$ URLs, originally ordered as $\{1, 2, 3, ..., n\}$. Suppose a classifier assigns a relevance level $\hat{y}_i \in \{0, 1, 2, 3, 4\}$ to the $i$th URL, for all $n$ URLs. A permutation mapping $\pi$ ranks the URLs according to $\hat{y}_i$, i.e., $\pi(i) < \pi(j)$ if $\hat{y}_i > \hat{y}_j$, and, URL $i$ and URL $j$ are arbitrarily ranked if $\hat{y}_i = \hat{y}_j$. The corresponding DCG error is bounded by the square root of the classification error,*

$$\text{DCG}_g - \text{DCG}_\pi \le 15\sqrt{2} \left( \sum_{i=1}^n c_{[i]}^2 - n \prod_{i=1}^n c_{[i]}^{2/n} \right)^{1/2} \left( \sum_{i=1}^n 1_{y_i \ne \hat{y}_i} \right)^{1/2}. \tag{4}$$

**Proof:**
$$
\begin{aligned}
\text{DCG}_\pi &= \sum_{i=1}^n c_{[\pi_i]} \left( 2^{y_i} - 1 \right) = \sum_{i=1}^n c_{[\pi_i]} \left( 2^{\hat{y}_i} - 1 \right) + \sum_{i=1}^n c_{[\pi_i]} \left( 2^{y_i} - 2^{\hat{y}_i} \right) \\
&\ge \sum_{i=1}^n c_{[g_i]} \left( 2^{\hat{y}_i} - 1 \right) + \sum_{i=1}^n c_{[\pi_i]} \left( 2^{y_i} - 2^{\hat{y}_i} \right) \\
&= \sum_{i=1}^n c_{[g_i]} \left( 2^{y_i} - 1 \right) - \sum_{i=1}^n c_{[g_i]} \left( 2^{y_i} - 2^{\hat{y}_i} \right) + \sum_{i=1}^n c_{[\pi_i]} \left( 2^{y_i} - 2^{\hat{y}_i} \right) \\
&= \text{DCG}_g + \sum_{i=1}^n \left( c_{[\pi_i]} - c_{[g_i]} \right) \left( 2^{y_i} - 2^{\hat{y}_i} \right).
\end{aligned}
$$

*Note that $\sum_{i=1}^{n} c_{[\pi_i]} \left(2^{\hat{y}_i} - 1\right) \geq \sum_{i=1}^{n} c_{[g_i]} \left(2^{\hat{y}_i} - 1\right)$. Therefore,*

$$DCG_g - DCG_\pi \leq \sum_{i=1}^{n} \left(c_{[g_i]} - c_{[\pi_i]}\right) \left(2^{y_i} - 2^{\hat{y}_i}\right)$$

$$\leq \left(\sum_{i=1}^{n} \left(c_{[g_i]} - c_{[\pi_i]}\right)^2\right)^{1/2} \left(\sum_{i=1}^{n} \left(2^{y_i} - 2^{\hat{y}_i}\right)^2\right)^{1/2} \leq \left(2\sum_{i=1}^{n} c_{[i]}^2 - 2n\prod_{i=1}^{n} c_{[i]}^{2/n}\right)^{1/2} 15 \left(\sum_{i=1}^{n} 1_{y_i \neq \hat{y}_i}\right)^{1/2}$$

*Note that $\sum_{i=1}^{n} c_{[\pi_i]}^2 = \sum_{i=1}^{n} c_{[g_i]}^2 = \sum_{i=1}^{n} c_{[i]}^2$, $\prod_{i=1}^{n} c_{[\pi_i]}^2 = \prod_{i=1}^{n} c_{[g_i]}^2 = \prod_{i=1}^{n} c_{[i]}^2$, and $2^4 - 2^0 = 15$.*

Thus, we can minimize the classification error $\sum_{i=1}^{n} 1_{y_i \neq \hat{y}_i}$ as a surrogate for minimizing the DCG error. Of course, since the classification error itself is non-convex and non-smooth, we actually should use other (well-known) surrogate loss functions such as (7).

## 3.2   Input Data for Classification

A training dataset contains $N_Q$ queries. The $j$th query corresponds to $n_j$ URLs; each URL is manually labeled by one of the $K = 5$ relevance levels. Engineers have developed methodologies to construct "features" by combining the query and URLs, but the details are usually "trade secret."

One important aspect in designing features, at least for the convenience of using traditional machine learning algorithms, is that these features should be comparable across queries. For example, one (artificial) feature could be the number of times the query appears in the Web page, which is comparable across queries. Both pair-based rankers and regression-based rankers implicitly made this assumption, as they tried to learn a single rank function for all queries using the same set of features.

Thus, after we have generated feature vectors by combining the queries and URLs, we can create a "training data matrix" of size $N \times P$, where $N = \sum_{j=1}^{N_Q} n_j$ is the total number of "data points" (i.e., Query+URL) and $P$ is the total number of features. This way, we can use the traditional machine learning notation $\{y_i, \mathbf{x}_i\}_{i=1}^{N}$ to denote the training dataset. Here $\mathbf{x}_i \in \mathbf{R}^P$ is the $i$th feature vector in $P$ dimensions; and $y_i \in \{0, 1, 2, 3, 4 = K - 1\}$ is the class (relevance) label of the $i$th data point.

## 3.3   From Classification to Ranking

Although perfect classifications lead to perfect DCG scores, in reality, we will need a mechanism to convert (imperfect) classification results into ranking scores.

One possibility is already mentioned in Lemma 1. That is, we classify each data point into one of the $K = 5$ classes and rank the data points according to the class labels (data points with the same labels are arbitrarily ranked). This suggestion, however, will lead to highly unstable ranking results.

Our proposed solution is very simple. We first learn the class probabilities by some *soft classification* algorithm and then score each data point (query+URL) according to the *Expected Relevance*.

Recall we assume a training dataset $\{y_i, \mathbf{x}_i\}_{i=1}^{N}$, where the class label $y_i \in \{0, 1, 2, 3, 4 = K - 1\}$. We learn the class probabilities $p_{i,k} = \mathbf{Pr}(y_i = k)$, denoted by $\hat{p}_{i,k}$, and define a scoring function:

$$S_i = \sum_{k=0}^{K-1} \hat{p}_{i,k} T(k), \tag{5}$$

where $T(k)$ is some monotone (increasing) function of the relevance level $k$. Once we have computed the scores $S_i$ for all data points, we can then sort the data points within each query by the descending order of $S_i$. This approach is apparently sensible and highly stable. In fact, we experimented with both $T(k) = k$ and $T(k) = 2^k$; the performance difference in terms of the NDCG scores was negligible, although $T(k) = k$ appeared to be a slightly better choice (see Figure 3(c) in Appendix II). In this paper, the reported experimental results were based on $T(k) = k$.

When $T(k) = k$, the scoring function $S_i$ is the *Expected Relevance*. Note that any monotone transformation on $S_i$ (e.g., $2^{S_i} - 1$) will not change the ranking results. Consequently, the ranking results are not affected by any affine transformation on $T(k)$, $aT(k) + b$, $(a > 0)$, because

$$\sum_{k=0}^{K-1} p_{i,k} \left(a \times T(k) + b\right) = a \times \left(\sum_{k=0}^{K-1} p_{i,k} T(k)\right) + b, \qquad \text{since} \sum_{k=0}^{K-1} p_{i,k} = 1. \tag{6}$$

### 3.4  The Boosting Tree Algorithm for Learning Class Probabilities

For multiple classification, we consider the following common (e.g., [8,9]) surrogate loss function

$$\sum_{i=1}^{N} \sum_{k=0}^{K-1} -\log(p_{i,k})1_{y_i=k}. \tag{7}$$

Algorithm 1 implements a boosting tree algorithm for learning class probabilities $p_{i,k}$; and we use basically the same implementation later for regression as well as multiple ordinal classification.

---

**Algorithm 1** The boosting tree algorithm for multiple classification, taken from [9, Algorithm 6], although the presentation is slightly different.

---

0: $\tilde{y}_{i,k} = 1$, if $y_i = k$, and $\tilde{y}_{i,k} = 0$ otherwise.
1: $F_{i,k} = 0$, $k = 0$ to $K - 1$, $i = 1$ to $N$
2: For $m = 1$ to $M$ Do
3:      For $k = 0$ to $K - 1$ Do
4:          $p_{i,k} = \exp(F_{i,k}) / \sum_{s=0}^{K-1} \exp(F_{i,s})$
5:          $\{R_{j,k,m}\}_{j=1}^{J} = J$-terminal node regression tree for $\{\tilde{y}_{i,k} - p_{i,k}, \; \mathbf{x}_i\}_{i=1}^{N}$
6:          $\beta_{j,k,m} = \frac{K-1}{K} \frac{\sum_{\mathbf{x}_i \in R_{j,k,m}} \tilde{y}_{i,k} - p_{i,k}}{\sum_{\mathbf{x}_i \in R_{j,k,m}} (1-p_{i,k})p_{i,k}}$
7:          $F_{i,k} = F_{i,k} + \nu \sum_{j=1}^{J} \beta_{j,k,m} 1_{\mathbf{x}_i \in R_{j,k,m}}$
8:      End
9: End

---

There are three main parameters. $M$ is the total number of boosting iterations, $J$ is the tree size (number of terminal nodes), and $\nu$ is the shrinkage coefficient. As commented in [9] and verified in our experiments, the performance of the algorithm is not sensitive to these parameters.

In Algorithm 1, Line 5 contains most of the implementation work, i.e., building the regression trees with $J$ terminal nodes. Appendix I describes an efficient implementation for building the trees.

## 4   Multiple Ordinal Classification to Further Improve Ranking

There is the possibility to (slightly) further improve our classification-based ranking scheme by taking into account the natural orders among the class labels, i.e., the multiple ordinal classification.

A common approach for multiple ordinal classification is to learn the cumulative probabilities $\mathbf{Pr}\,(y_i \leq k)$ instead of the class probabilities $\mathbf{Pr}\,(y_i = k) = p_{i,k}$. We suggest a simple method similar to the so-called cumulative logits approach known in statistics [1, Section 7.2.1].

We first partition the training data points into two groups: $\{y_i \geq 4\}$ and $\{y_i \leq 3\}$. Now we have a binary classification problem and hence we can use exactly the same boosting tree algorithm for multiple classification. Thus we can learn $\mathbf{Pr}\,(y_i \leq 3)$ easily. We can similarly partition the data and learn $\mathbf{Pr}\,(y_i \leq 2)$, $\mathbf{Pr}\,(y_i \leq 1)$, and $\mathbf{Pr}\,(y_i \leq 0)$, separately. We then infer the class probabilities

$$p_{i,k} = \mathbf{Pr}\,(y_i = k) = \mathbf{Pr}\,(y_i \leq k) - \mathbf{Pr}\,(y_i \leq k-1), \tag{8}$$

and again we use the *Expected Relevance* to compute the ranking scores and sort the URLs.

We call both rankers based on multiple classification and multiple ordinal classification as *McRank*.

## 5   Regression-based Ranking Using Boosting Tree Algorithm

With slight modifications, the boosting tree algorithm can be used for regressions. Recall the input data are $\{y_i, \mathbf{x}_i\}_{i=1}^{N}$, where $y_i \in \{0, 1, 2, 3, 4\}$. [6] suggested regressing the feature vectors $\mathbf{x}_i$ on the response values $2^{y_i} - 1$.

Algorithm 2 implements the least-square boosting tree algorithm. The pseudo code is similar to [9, Algorithm 3] by replacing the ($l_1$) least absolute deviation (LAD) loss with the ($l_2$) least square loss. In fact, we also implemented the LAD boosting tree algorithm but we found the performance was considerably worse than the least-square tree boost.

**Algorithm 2** The boosting tree algorithm for regressions. After we have learned the values for $S_i$, we use them directly as the ranking scores to order the data points within each query.

0: $\tilde{y}_i = 2^{y_i} - 1$
1: $S_i = \frac{1}{N} \sum_{s=1}^{N} \tilde{y}_s, \ i = 1$ to $N$
2: For $m = 1$ to $M$ Do
5:        $\{R_{j,m}\}_{j=1}^{J} = J$-terminal node regression tree for $\{\tilde{y}_i - S_i, \ \mathbf{x}_i\}_{i=1}^{N}$
6:        $\beta_{j,m} = \text{mean}_{\mathbf{x}_i \in R_{j,m}} \tilde{y}_i - S_i$
7:        $S_i = S_i + \nu \sum_{j=1}^{J} \beta_{j,m} 1_{\mathbf{x}_i \in R_{j,m}}$
9: End

## 6 Experimental Results

We present the evaluations of 4 ranking algorithms (*LambdaRank* with two-layer nets, regression, multiple classification, and multiple ordinal classification) on 4 datasets, including one artificial dataset and three Web search datasets, denoted by Web-1, Web-2, and Web-3. The artificial dataset and Web-1 are the same datasets used in [5]. Web-2 is the main dataset used in [13].

For the artificial data and Web-1, [5] reported that *LambdaRank* improved *RankNet* by about 1.0 (%) NDCG. For Web-2, [13] reported that *FRank* slightly improved *RankNet* (by about 0.5 (%) NDCG) and considerably improved *RankSVM* and *RankBoost*; but [13] did not compare with *LambdaRank*. Our experiment showed that *LambdaRank* improved *FRank* by about 0.9 (%) NDCG on Web-2.

### 6.1 The Datasets

The artificial dataset [5] was meant to remove any variance caused by the quality of features and/or relevance labels. The data were generated from random cubic polynomials, with 50 features, 50 URLs per query, and 10,000/5,000/10,000 queries for train/validation/test.

The Web search dataset Web-1 [5] has 367 features and 10,000/5,000/10,000 queries for train/validation/test, with in total 652,500 URLs.

Web-2 [13] has 619 features and 12,000/3,800/3,800 queries for train/validation/test, with in total 1,741,930 URLs. Note that this dataset is only partially labeled with 20 unlabeled URLs per query. These unlabeled URLs were assigned the level 0 [13].

Web-3 has 450 features and 26,000 queries, with in total 474,590 URLs. We conducted five-fold cross-validations and report the average NDCG scores.

### 6.2 The Parameters: $M$, $J$, $\nu$

There are three main parameters in the boosting tree algorithm. $M$ is the total number of iterations, $J$ is the number of terminal nodes in each tree, and $\nu$ is the shrinkage factor. Our experiments verify that these parameters are not sensitive as long as they are within some "reasonable" ranges [9]. Since these experiments are time-consuming, we did not tune these parameters ($M$, $J$, $\nu$) exhaustively; but the experiments appear to be convincing enough to establish the superiority of *McRank*.

[9] suggested setting $\nu \leq 0.1$, to avoid over-fitting. We fix $\nu = 0.05$ for the artificial dataset and Web-1, and fix $\nu = 0.02$ for Web-2 and Web-3. The number of terminal nodes, $J$, should be reasonably big (but not too big) when the dataset is large with a large number of features, because the tree has to be deep enough to consider higher-order interactions [9]. We let $J = 10$ for the artificial dataset and Web-1, $J = 40$ for Web-2, and $J = 20$ for Web-3.

With these values of $J$ and $\nu$, we did not observe obvious over-fitting even for a very large number of boosting iterations $M$. We will report the results with $M = 1000$ for the artificial data and Web-1, $M = 2000$ for Web-2, and $M = 1500$ for Web-3.

### 6.3 The Test NDCG Results at Truncation Level $L = 10$

Table 1 lists the NDCG results (both the mean and standard deviation, in percentages (%)) for all 4 datasets and all 4 ranking algorithms, evaluated at the truncation level $L = 10$.

The NDCG scores indicate that that *McRank* (ordinal classification and classification) considerably improves the regression-based ranker and *LambdaRank*. If we conduct a one-sided $t$-test, the im-

Table 1: The test NDCG scores produced by 4 rankers on 4 datasets. The average NDCG scores are presented in percentages (%) with the standard deviations in the parentheses. Note that for the artificial data and Web-1, the *LambdaRank* results were taken directly from [5]. We also report the (one-sided) $p$-values to measure the statistical significance of the improvement of *McRank* over regression and *LambdaRank*. For the artificial data, Web-1, and Web-3, we use the ordinal classification results to compute the $p$-values. However, for Web-2, because our implementation for testing ordinal classification required too much memory for $M = 2000$, we did not obtain the final test NDCG scores; the partial results indicated that ordinal classification did not improve classification for this dataset. Therefore, we compute the $p$-values using classification results for Web-2.

| Datasets | Ordinal Classification | Classification | Regression, $p$-value | LambdaRank, $p$-value |
|---|---|---|---|---|
| Artificial [5] | 85.0 (9.5) | 83.7 (9.9) | 82.9 (10.2), 0 | 74.9, (12.6), 0 |
| Web-1 [5] | 72.4 (24.1) | 72.2 (24.1) | 71.7 (24.4), 0.021 | 71.2 (24.5), 0.0002 |
| Web-2 [13] | — | 75.8 (23.8) | 74.7 (24.4), 0.023 | 74.3 (24.3), 0.003 |
| Web-3 | 72.5 (26.5) | 72.4 (27.3) | 72.0 (27.6), 0.017 | 71.3 (28.8), $3.8 \times 10^{-7}$ |

provements are significant at about $98\%$ level. However, multiple ordinal classification did not show significant improvement over multiple classification, except for the artificial dataset.

For the artificial data, all other 3 rankers exhibit very large improvements over *LambaRank*. This is probably due to the fact that the artificial data are generated noise-free and hence the flexible (with high capacity) rankers using boosting tree algorithms tend to fit the data very well.

### 6.4 The NDCG Results at Various Truncation Levels ($L = 1$ to $10$)

For the artificial dataset and Web-1, [5] also reported the NDCG scores at various truncation levels, $L = 1$ to $10$. To make the comparisons more convincing, we also report similar results for the artificial dataset and Web-1, in Figure 1. For a better clarity, we plot the standard deviations separately from the averages. Figure 1 verifies that the improvements shown in Table 1 are not only true for $L = 10$ but also (essentially) true for smaller truncation levels.

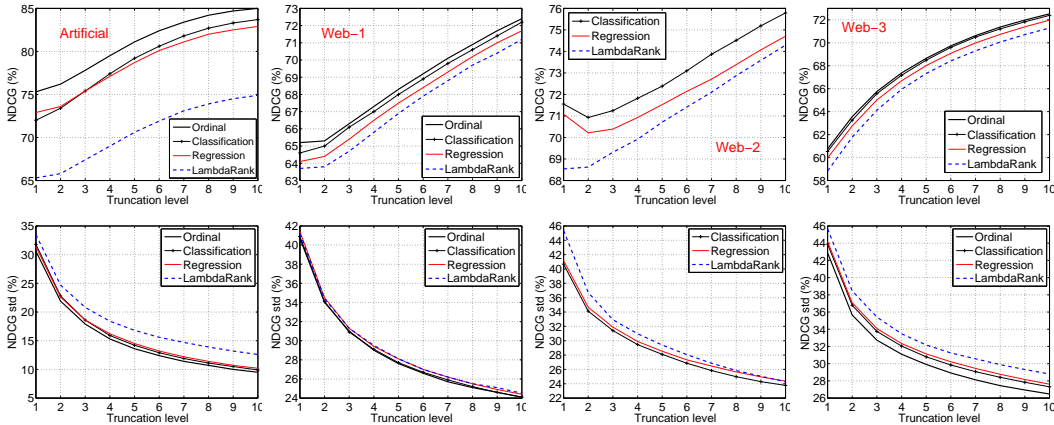

Figure 1: The NDCG scores at truncation levels $L = 1$ to $10$, for four datasets. Upper Panels: the average NDCG scores. Bottom Panels: the corresponding standard deviations.

## 7 Conclusion

The ranking problem has become an important topic in machine learning, partly due to its widespread applications in many decision-making processes especially in commercial search engines. In one aspect, the ranking problem is difficult because the measures of rank quality are usually based on sorting, which is not directly optimizable (at least not efficiently). On the other hand, one can cast ranking into various classical learning tasks such as regression and classification.

The proposed classification-based ranking scheme is motivated by the fact that perfect classifications lead to perfect DCG scores and the DCG errors are bounded by the classification errors. It appears

natural that the classification-based ranker is more direct and should work better than the regression-based ranker suggested in [6]. To convert classification results into ranking, we propose a simple and stable mechanism by using the *Expected Relevance*, computed from the learned class probabilities.

To learn the class probabilities, we implement a boosting tree algorithm for multiple classification and we use the same implementation for multiple ordinal classification and regression. Since commercial proprietary datasets are usually very large, an adaptive quantization-based approach efficiently implements the boosting tree algorithm, which avoids sorting and has lower memory cost.

Our experimental results have demonstrated that *McRank* (including multiple classification and multiple ordinal classification) outperforms both the regression-based ranker and the pair-based *LambdaRank*. However, except for the artificial dataset, we did not observe significant improvement of ordinal classification over classification.

In a summary, we regard *McRank* algorithm (retrospectively) simple, robust, and capable of producing quality ranking results.

# Appendix I    An Efficient Implementation for Building Boosting Trees

We use the standard regression tree algorithm [2], which recursively splits the training data points into two groups on the current "best" feature that will reduce the mean square errors (MSE) the most. Efficient (in both time and memory) implementation needs some care. The standard practice [9] is to pre-sort all the features; then after every split, carefully keep track of the indexes of the data points and the sorted orders in all other features for the next split.

We suggest a simpler and more efficient approach, by taking advantage of some properties of the boosting tree algorithm. While the boosting tree algorithm is well-known to be robust and also accurate, an individual tree has limited predictive power and usually can be built quite crudely.

When splitting on one feature, Figure 2(a) says that sometimes the split point can be chosen within a certain range without affecting the accuracy (i.e., the reduced MSE due to the split). In Figure 2(b), we bin (quantize) the data points into two (0/1) levels on the horizontal (i.e., feature) axis. Suppose we choose the quantization as shown in the Figure 2(b), then the accuracy will not be affected either.

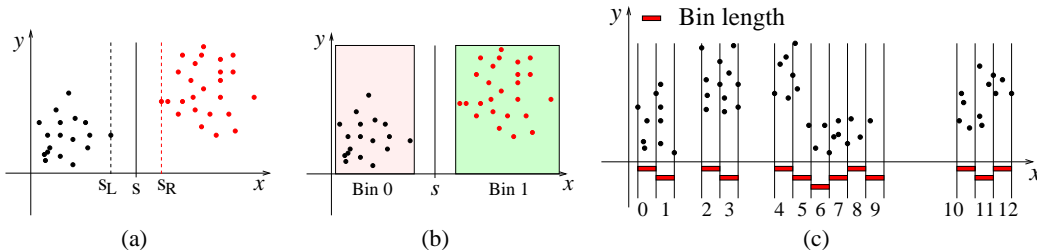

Figure 2: To split on one feature ($x$), we seek a split point $s$ on $x$ such that after the splitting, the mean square error (MSE, in the $y$ axis) of the data points at the left plus the MSE at the right is reduced the most. Panel (a) suggests that in some cases we can choose $s$ in a range (within $s_L$ and $s_R$) without affecting the reduced MSE. Panel (b) suggests that, if we bin the data on the $x$ axis to be binary, the reduced MSE will not be affected either, if the data are binned in the way as in (b). Panel (c) pictures an adaptive binning scheme to make the accuracy loss (if any) as little as possible.

Of course, we would not know ahead of time how to bin the data to avoid losing accuracy. Therefore, we suggest an adaptive quantization scheme, pictured in Figure 2(c), to make the accuracy loss (if any) as little as possible. In the pre-processing stage, for each feature, the training data points are sorted according to the feature value; and we bin the feature values in the sorted order. We start with a very small initial bin length, e.g., $10^{-8}$. As shown in Figure 2(c), we only bin the data where there are indeed data, because the boosting tree algorithm will not consider the area where there are no data anyway. We set an allowed maximum number of bins, denoted by $B$. If the bin length is so small that we need more than $B$ bins, we simply increment the bin length and re-do the quantization. After the quantization, we replace the original feature value by the bin labels (0, 1, 2, ...). Note that since we start with a small bin length, the ordinal categorical features are naturally taken care of.

This simple binning scheme is very effective particularly for the boosting tree algorithm:

- It simplifies the implementation. After the quantization, there is no need for sorting (and keeping track of the indexes after splitting) because we conduct "bucket sort" implicitly.

- It speeds up the computations for the tree-building step, the bottleneck of the algorithm.

- It reduces the memory cost for training. For example, if we set the maximum allowed number of bins to be $B = 2^8$, we only need one byte per data entry.

- It does not really result in loss of accuracy. We experimented with both $B = 2^8 = 256$ and $B = 2^{16} = 65536$; and we did not observe real differences in the NDCG scores, although reported experimental results were all based on $B = 2^{16}$. See Appendix II, Figure 3(a)(b).

## Appendix II    Some More Experiments on Web-1

Figure 3 (a)(b) present the experiment with our adaptive quantization scheme on Web-1 dataset. We binned the data with the maximum bin number $B = 2^3, 2^4, 2^5, 2^6, 2^7, 2^8$, and $2^{16}$. In (a) and (b), the horizontal axis is the "exponent" of $B$. Panel (a) plots the relative number of total bins in Web-1 as a function of the exponent, normalized by the total number of bins at $B = 2^{16}$. Panel (b) plots the "NDCG loss" due to the quantization, relative to the NDCG scores at $B = 2^{16}$. When $B = 2^8$, the total number of bins is only about 6% of that when $B = 2^{16}$; however, both quantization levels achieved the same test NDCG scores. Besides the benefit of computational efficiency, quantization can also be considered as a way of "regularization" to slow down the training, as reflected in (b).

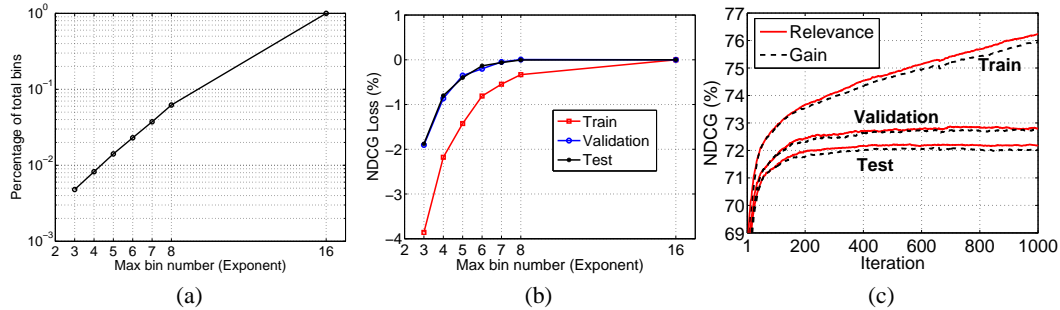

Figure 3: Web-1. (a)(b): Experiment with our adaptive quantization scheme. (c): Experiment with two different scoring functions.

Figure 3 (c) compares two scoring functions to convert learned class probabilities into ranking scores, including the *Expected Relevance* $S_i = \sum_{k=0}^{K-1} \hat{p}_{i,k} k$ and the *Expected Gain* $S_i = \sum_{k=0}^{K-1} \hat{p}_{i,k} \left( 2^k - 1 \right)$. Panel (c) suggests that using the *Expected Relevance* is consistently better than using the *Expected Gain* but the differences are small, especially for the test NDCG scores.

## Footnotes

*Much of the work was conducted while Ping Li was an intern at Microsoft in 2006.

[1]In fact LambdaRank supports any preference function, although the reported results in [5] are for pairwise.

## References

[1]  A. Agresti. *Categorical Data Analysis*. John Wiley & Sons, Inc., Hoboken, NJ, second edition, 2002.

[2]  L. Brieman, J. Friedman, R. Olshen, and C. Stone. *Classification and Regression Trees*. 1983.

[3]  S. Brin and L. Page. The anatomy of a large-scale hypertextual web search engine. In *WWW*, pages 107–117, 1998.

[4]  C. Burges, T. Shaked, E. Renshaw, A. Lazier, M. Deeds, N. Hamilton, and G. Hullender. Learning to rank using gradient descent. In *ICML*, pages 89–96, 2005.

[5]  C. Burges, R. Ragno, and Q. Le. Learning to rank with nonsmooth cost functions. In *NIPS*, pages 193–200, 2007.

[6]  D. Cossock and T. Zhang. Subset ranking using regression. In *COLT*, pages 605–619, 2006.

[7]  Y. Freund, R. Iyer, R. Schapire, and Y. Singer. An efficient boosting algorithm for combining preferences. *Journal of Machine Learning Research*, 4:933–969, 2003.

[8]  J. Friedman, T. Hastie, and R. Tibshirani. Additive logistic regression: a statistical view of boosting. *The Annals of Statistics*, 28(2):337–407, 2000.

[9]  J. Friedman. Greedy function approximation: A gradient boosting machine. *The Annals of Statistics*, 29(5):1189–1232, 2001.

[10]  K. Järvelin and J. Kekäläinen. IR evaluation methods for retrieving highly relevant documents. In *SIGIR*, pages 41–48, 2000.

[11]  T. Joachims. Optimizing search engines using clickthrough data. In *KDD*, pages 133–142, 2002.

[12]  J. Kleinberg. Authoritative sources in a hyperlinked environment. In *SODA*, pages 668–677, 1998.

[13]  M. Tsai, T. Liu, T. Qin, H. Chen, and W. Ma. Frank: a ranking method with fidelity loss. In *SIGIR*, pages 383–390, 2007.

[14]  Z. Zheng, K. Chen, G. Sun, and H. Zha. A regression framework for learning ranking functions using relative relevance judgments. In *SIGIR*, pages 287-294, 2007.
